# Structure Learning in Human Sequential Decision-Making

**Daniel Acuña**
Dept. of Computer Science and Eng.
University of Minnesota–Twin Cities
acuna002@umn.edu

**Paul Schrater**
Dept. of Psychology and Computer Science and Eng.
University of Minnesota–Twin Cities
schrater@umn.edu

## Abstract

We use graphical models and structure learning to explore how people learn policies in sequential decision making tasks. Studies of sequential decision-making in humans frequently find suboptimal performance relative to an ideal actor that knows the graph model that generates reward in the environment. We argue that the learning problem humans face also involves learning the graph structure for reward generation in the environment. We formulate the structure learning problem using mixtures of reward models, and solve the optimal action selection problem using Bayesian Reinforcement Learning. We show that structure learning in one and two armed bandit problems produces many of the qualitative behaviors deemed suboptimal in previous studies. Our argument is supported by the results of experiments that demonstrate humans rapidly learn and exploit new reward structure.

## 1 Introduction

Humans daily perform sequential decision-making under uncertainty to choose products, services, careers, and jobs; and to mate and survive as species. One of the central problems in sequential decision making with uncertainty is balancing exploration and exploitation in the search for good policies. Using model-based (Bayesian) Reinforcement learning [1], it is possible to solve this problem optimally by finding policies that maximize the expected discounted future reward [2]. However, solutions are notoriously hard to compute, and it is unclear whether optimal models are appropriate for human decision-making. For tasks simple enough to allow comparison between human behavior and normative theory, like the multi-armed bandit problem, human choices appear suboptimal. In particular, earlier studies suggested human choices reflect inaccurate Bayesian updating with sub-optimalities in exploration [3, 4, 5, 6]. Moreover, in one-armed bandit tasks where exploration is not necessary, people frequently converge to probability matching [7, 8, 9, 10], rather than the better option, even when subjects are aware which option is best [11]. However, failures against normative prediction may reflect optimal decion-making, but for a task that differs from the experimenter's intention. For example, people may assume the environment is potentially dynamically varying. When this assumption is built into normative predictions, these models account much better for human choices in one-armed bandit problems [12], and potentially multi-armed problems [13]. In this paper, we investigate another possibility, that humans may be learning the structure of the task by forming beliefs over a space of canonical causal models of reward-action contingencies.

Most human performance assessments view the subject's task as parameter estimation (e.g. reward probabilities) within a known model (a fixed causal graph structure) that encodes the relations between environmental states, rewards and actions created by the experimenter. However despite instruction, it is reasonable that subjects may be uncertain about the model, and instead try to learn it. To illustrate structure learning in a simple task, suppose you are alone in a casino with many rooms. In one room you find two slot machines. It is typically assumed you know the machines are

independent and give rewards either 0 (failure) or 1 (success) with unknown probabilities that must be estimated. The structure learning viewpoint allows for more possibilities: Are they independent, or are are they rigged to covary? Do they have the same probability? Does reward accrue when the machine is not played for a while? We believe answers to these questions form a natural set of causal hypotheses about how reward/action contingencies may occur in natural environments.

In this work, we assess the effect of uncertainty between two critical reward structures in terms of the need to explore. The first structure is a one-arm bandit problem in which exploration is not necessary (reward generation is coupled across arms); greedy action is optimal [14]. And the other structure is a two-arm bandit problem in which exploration is necessary (reward generation is independent at each arm); each action needs to balance the exploration/exploitation tradeoff [15]. We illustrate how structure learning affects action selection and the value of information gathering in a simple sequential choice task resembling a Multi-armed Bandit (MAB), but with uncertainty between the two previous models of reward coupling. We develop a normative model of learning and action for this class of problems, illustrate the effect of model uncertainty on action selection, and show evidence that people perform structure learning.

## 2 Bayesian Reinforcement Learning: Structure Learning

The language of graphical models provides a useful framework for describing the possible structure of rewards in the environment. Consider an environment with several distinct reward sites that can be sampled, but the way models generate these rewards is unknown. In particular, rewards at each site may be independent, or there may be a latent cause which accounts for the presence of rewards at both sites. Even if independent, if the reward sites are homogeneous, then they may have the same probability.

Uncertainty about which reward model is correct naturally produces a mixture as the appropriate learning model. This structure learning model is a special case of Bayesian Reinforcement Learning (BRL), where the states of the environment are the reward sites and the transitions between states are determined by the action of sampling a reward site. Uncertainty about reward dynamics and contingencies can be modeled by including within the belief state not only reward probabilities, but also the possibility of independent or coupled rewards. Then, the optimal balance of exploration and exploitation in BRL results in action selection that seeks to maximize (1) expected rewards (2) information about rewards dynamics, and (3) information about task structure.

Given that tasks tested in this research involve mixtures of Multi-Armed Bandit (MAB) problems, we borrow MAB language to call a reward site, an arm, and sample a choice or pull. However, the mixture models we describe are not MAB problems. MAB problems require the dynamics of one site (arm) remain *frozen* until visited again, which is not true in general for our mixture model.

Let $\gamma$ ($0 < \gamma < 1$) be a discounting factor such that a possibly stochastic reward $x$ obtained $t$ time steps in the future means $\gamma^t x$ today. Optimality requires an action selection policy that maximizes the expectation over the total discounted future reward $\mathbb{E}_b\left[x + \gamma x + \gamma^2 x + \dots\right]$, where $b$ is the belief over environment dynamics. Let $x_a$ be a reward acquired from arm $a$. After observing reward $x_a$, we compute a belief state posterior $b_{xa} \equiv p(b|x_a) \propto p(x_a|b)p(b)$. Let $f(x_a|b) \equiv \int db\, p(x_a|b)p(b)$ be the predicted probability of reward $x_a$ given belief $b$. Let $r(b,a) \equiv \sum x_a f(x_a \mid b)$ be the expected reward of sampling arm $a$ at state $b$. The *value* of a state can be found using the Bellman equation [2],

$$V(b) = \max_a \left\{ r(b,a) + \gamma \sum_{x_a} f(x_a \mid b) V(b_{xa}) \right\}. \tag{1}$$

The optimal action can be recovered by choosing arm

$$a = \arg\max_{a'} \left\{ r(b,a') + \gamma \sum_{x_a} f(x_{a'} \mid b) V(b_{xa'}) \right\}. \tag{2}$$

The belief over dynamics is effectively a probability distribution over possible Markov Decision Processes that would explain observables. As such, the optimal policy can be described as a mapping from belief *states* to actions. In principle, the optimal solution can be found by solving Bellman optimality equations but generally there are countably or uncountably infinitely many states and solutions need approximations.

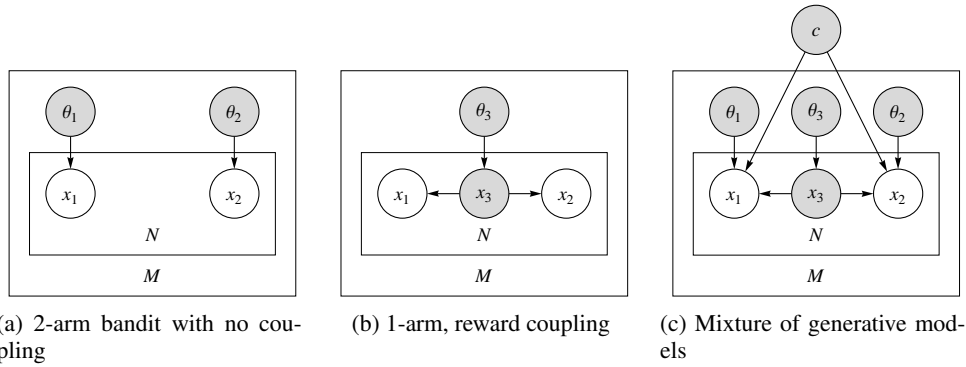

(a) 2-arm bandit with no coupling

(b) 1-arm, reward coupling

(c) Mixture of generative models

Figure 1: Different graphical models for generation of rewards at two known sites in the environment. The agent faces $M$ bandit tasks each comprising a random number of $N$ choices **(a)** Reward sites are independent. **(b)** Rewards are dependent within a bandit task **(c)** Mixture of generative models used by the learning model. The causes of reward may be independent or coupled. The node $c$ acts as a "XOR" switch between coupled and independent reward.

In Figure 1, we show the two reward structures considered on this paper. Figure 1(a) illustrates a structure where arms are independent and (b) coupled. When independent, rewards $x_a$ at arm $a$ are samples from a unknown distribution $p(x_a|\theta_a)$. When coupled, rewards $x_a$ depends on a "hidden" state of reward $x_3$ sampled from $p(x_3|\theta_3)$. In this case, the rewards $x_1$ and $x_2$ are coupled and depends on $x_3$.

If we were certain which of the two models were right, the action selection problem has known solution for both cases, presented below.

**Independent Rewards**. Learning and acting in an environment like the one described in Figure 1(a) is known as the Multi-Armed Bandit (MAB) problem. The MAB problem is a special case of BRL because we can partition the belief $b$ into a disjoint set of beliefs about each arm $\{b_a\}$. Because beliefs about non-sampled arms remain *frozen* until sampled again and sampling one arm doesn't affect the belief about any other, independent learning and action selection for each arm is possible. Let $\lambda_a$ be the reward of a deterministic arm in $V(b_a) = \max\{\lambda_a/(1-\gamma), r(b_a,a) + \gamma\sum f(x_a|b_a)V(b_{xa})\}$ such that both terms inside the maximization are equal. Gittins [16] proved that it is optimal to choose the arm $a$ with the highest such reward $\lambda_a$ (called the Gittins Index). This allows speedup of computation by transforming a *many*-arm bandit problem to *many* 2-arm bandit problems.

In our task, the belief about a binary reward may be represented by a Beta Distribution with sufficient statistics parameters $\alpha, \beta$ (both $> 0$) such that $x_a \sim p(x_a|\theta_a) = \theta_a^{x_a}(1-\theta_a)^{1-x_a}$, where $\theta_a \sim p(\theta_a; \alpha_a, \beta_a) \propto \theta_a^{\alpha_a-1}(1-\theta_a)^{\beta_a-1}$. Thus, the expected reward $r(\alpha_a, \beta_a, a)$ and predicted probability of reward $f(x_a = 1|\alpha_a, \beta_a)$ are $\alpha_a(\alpha_a + \beta_a)^{-1}$. The belief state transition is $b_{xa} = \langle \alpha_a + x_a, \beta_a + 1 - x_a \rangle$. Therefore, the Gittins index may be found by solving the Bellman equations using dynamic programming $V(\alpha_a, \beta_a) = \max\{\lambda_a(1-\gamma)^{-1}, (\alpha_a+\beta_a)^{-1}[\alpha_a + \gamma(\alpha_a + \alpha_a V(\alpha_a+1, \beta_a) + \beta_a V(\alpha_a, \beta_a+1))]\}$ to a sufficiently large horizon. In experiments, we use $\gamma = 0.98$, for which a horizon of $H = 1000$ suffices.

**Coupled Rewards.** Learning and acting in coupled environments (Figure 1b) is trivial because there is no need to maximize information in acting [14]. The belief state is represented by a Beta distribution with sufficient statistics $\alpha_3, \beta_3$ $(> 0)$. Therefore, the optimal action is to choose the arm $a$ with highest expected reward

$$r(\alpha_3, \beta_3, a) = \begin{cases} \frac{\alpha_3}{\alpha_3+\beta_3} & a=1 \\ \frac{\beta_3}{\alpha_3+\beta_3} & a=2 \end{cases}$$

The belief state transitions are $b_1 = \langle \alpha_3 + x_1, \beta_3 + 1 - x_1 \rangle$ and $b_2 = \langle \alpha_3 + 1 - x_2, \beta_3 + x_2 \rangle$.

# 3 Learning and acting with model uncertainty

In this section, we consider the case where there is uncerainty about the reward model. The agent's belief is captured by a graphical model for a family of reward structures that may or may not be coupled. We show that learning can be accurate and that action selection is relatively efficient.

We restrict ourselves to the following scenario. The agent is presented with a block of $M$ bandit tasks, each with initially unknown Bernoulli reward probabilities and coupling. Each task involves $N$ discrete choices, where $N$ is sampled from a Geometric distribution $(1-\gamma)\gamma^N$.

Figure 1(c) shows the mixture of two possible reward models shown in figure 1(a) and (b). Node $c$ switches the mixture between the two possible reward models and encodes part of the belief state of the process. Notice that $c$ is acting as a "XOR" gate between the two generative models. Given that it is unknown, the probability distribution $p(c=0)$ is the mixed proportion for independent reward structure and $p(c=1)$ is the mixed proportion for coupled reward structure. We put a prior on the state $c$ using the distribution $p(c;\phi) = \phi^c(1-\phi)^{1-c}$, with parameter $\phi$. The posterior is

$$
\begin{aligned}
&p(\theta_1,\theta_2,\theta_3,c|s_1,f_1,s_2,f_2) = \\
&\propto \begin{cases} (1-\phi) \times \left( \theta_1^{\alpha_1-1+s_1}(1-\theta_1)^{\beta_1-1+f_1}\theta_2^{\alpha_2-1+s_2}(1-\theta_2)^{\beta_2-1+f_2}\theta_3^{\alpha_3-1}(1-\theta_3)^{\beta_3-1} \right) & c=0 \\ (\phi) \times (\theta_1^{\alpha_1-1}(1-\theta_1)^{\beta_1-1}\theta_2^{\alpha_2-1}(1-\theta_2)^{\beta_2-1}\theta_3^{\alpha_3-1+s_1+f_2}(1-\theta_3)^{\beta_3-1+s_2+f_1} & c=1 \end{cases}
\end{aligned}
\tag{3}
$$

where $s_a$ and $f_a$ is the number of successes and failures observed in arm $a$. It is clear that the posterior (3) is a mixture of the beliefs on parameters $\theta_j$, for $1 \leq j \leq 3$. With mixed proportion $\phi$, successes of arm 1 and failures of arm 2 are attributed to successes on the shared "hidden" arm 3, whereas failures of arm 1 and successes of arm 2 are attributed to failures of arm 3. On the other hand, the usual Beta-Bernoulli learning of independent arms happens with mixed proportion $1-\phi$.

At the beginning of each bandit task, we assume the agent "resets" its belief about arms ($s_i = f_i = 0$), but the posterior over $p(c)$ is carried over and used as the prior on the next bandit task. Let $\mathrm{Beta}(\alpha,\beta)$ be the Beta function. The marginal posterior on $c$ is as follows

$$
p(c|s_1,f_1,s_2,f_2) \quad \propto \quad \begin{cases} (1-\phi)\frac{\mathrm{Beta}(\alpha_1+s_1,\beta_1+f_1)\mathrm{Beta}(\alpha_2+s_2,\beta_2+f_2)}{\mathrm{Beta}(\alpha_1,\beta_1)\mathrm{Beta}(\alpha_2,\beta_2)} & c=0 \\ \phi\frac{\mathrm{Beta}(\alpha_3+s_1+f_2,\beta_3+f_1+s_2)}{\mathrm{Beta}(\alpha_3,\beta_3)} & c=1 \end{cases}
$$

The belief state $b$ of this process may be completely represented by $\langle s_1,f_1,s_2,f_2;\phi,\alpha_1,\beta_1,\alpha_2,\beta_2,\alpha_3,\beta_3\rangle$. The predicted probability of reward $x_1$ and $x_2$ are:

$$
f(x_1|s_1,f_1,s_2,f_2) = \begin{cases} (1-\phi)\frac{\alpha_1+s_1}{\alpha_1+s_1+\beta_1+f_1} + \phi\frac{\alpha_3+s_1+f_2}{\alpha_3+s_1+f_2+\beta_3+s_2+f_1} & x_1=1 \\ (1-\phi)\frac{\beta_1+f_1}{\alpha_1+s_1+\beta_1+f_1} + \phi\frac{\beta_3+s_2+f_1}{\alpha_3+s_1+f_2+\beta_3+s_2+f_1} & x_1=0 \end{cases}
\tag{4}
$$

and similarly

$$
f(x_2|s_1,f_1,s_2,f_2) = \begin{cases} (1-\phi)\frac{\alpha_2+s_2}{\alpha_2+s_2+\beta_2+f_2} + \phi\frac{\beta_3+s_2+f_1}{\alpha_3+s_1+f_2+\beta_3+s_2+f_1} & x_2=1 \\ (1-\phi)\frac{\beta_2+f_2}{\alpha_2+s_2+\beta_2+f_2} + \phi\frac{\alpha_3+s_1+f_2}{\alpha_3+s_1+f_2+\beta_3+s_2+f_1} & x_2=0 \end{cases}
\tag{5}
$$

Let us drop prior parameters $\alpha_j,\beta_j$, $1 \leq j \leq 3$, and $\phi$ from $b$. The action selection involves solving the following Bellman equations

$$
\begin{aligned}
&V(s_1,f_1,s_2,f_2) = \\
&\max_{a=1,2} \begin{cases} r(b,1) + \gamma[f(x_1=0|b)V(s_1,f_1+1,s_2,f_2) + f(x_1=1|b)V(s_1+1,f_1,s_2,f_2)] & a=1 \\ r(b,2) + \gamma[f(x_2=0|b)V(s_1,f_1,s_2,f_2+1) + f(x_2=1|b)V(s_1,f_1,s_2+1,f_2)] & a=2 \end{cases}
\end{aligned}
\tag{6}
$$

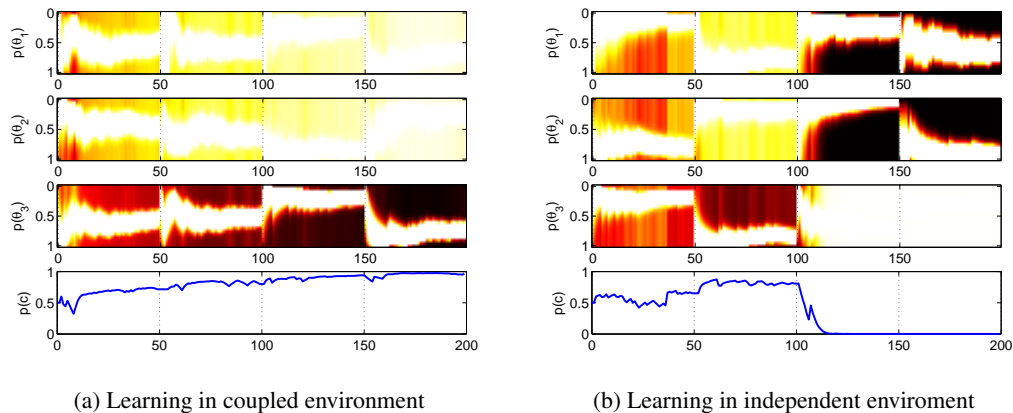

<div align="center">(a) Learning in coupled environment        (b) Learning in independent enviroment</div>

Figure 2: Learning example. A block of four bandit tasks of 50 trials each for each environment. Marginal beliefs on reward probabilities and coupling are shown as functions of time. The brightness indicates the relative probability mass. The coupling belief distribution starts uniform with $\phi = 0.5$ and is not reset within a block. The priors $p(\theta_i; \alpha_i, \beta_i)$ are reset at the beginning of each task with $\alpha_i, \beta_i = 1$ ($1 \leq i \leq 3$) . Note that how well the reward probabilities sum to one forms critical evidence for or against coupling.

To obtain (6) using dynamic programing for a horizon $H$, there will be a total of $(1/24)(1+H)(2+H)(3+H)(4+H)$ computations which represent different occurrences of $s_i, f_i$ out of $4^H$ possible histories of rewards. This dramatic reduction allows us to be relatively accurate in our approximation to the optimal value of an action.

## 4    Simulation Results

In Figure 2, we perform simulations of learning on blocks of four bandit tasks, each comprising 50 trials. In one simulation, (a) rewards are coupled and the other (b) independent. Note that the model learns quickly on both cases, but it is slower when task is truly coupled because fewer cases support this hypothesis (when compared to the independent hypothesis).

The importance of the belief on the coupling parameter is that it has a decisive influence on exploratory behavior. Coupling between the two arms corresponds to the case where one arm is a winner and the other is a loser by experimenter design. When playing coupled arms, evidence that an arm is "good" (e.g. $> 0.5$) necessarily entails the other is "bad", and hence eliminates the need for exploratory behavior - the optimal choice is to "stick with the winner", and switch when the probability estimate suggests dips below 0.5. An agent learning a coupling parameter while sampling arms can manifest a range of exploratory behaviors that depend critically on both the recent reward history and the current state of the belief about $c$, illustrated in figure 3. The top row shows the value of both arms as a function of coupling belief $p(c)$ after different amounts of evidence for the success of arm 2. The plots show that optimal actions stick with the winner when belief in coupling is high, even for small amounts of data. Thus belief in coupling produces underexploration compared to a model assuming independence, and generates behavior similar to a "win stay, lose switch" heuristic early in learning. However, overexploration can also occur when the expected values of both arms are similar. Figure 3 (lower left) shows that uncertainty about $c$ provides an exploratory bonus to the lower probability arm which incentivizes switching, and hence overexploration. In fact, when the difference in probability between arms is small, action selection can fail to converge to the better option. Figure 3 (to the right) shows that $p(c)$ together with the probability of the better arm determine the transition between exploration vs. exploitation. These results show that optimal action selection with model uncertainty can generate several kinds of behavior typically labeled suboptimal in multi-armed bandit experiments. Next we provide evidence that people are capable of learning and exploiting coupling–evidence that structure learning may play a role in apparent failures of humans to behave optimally in multi-armed bandit tasks.

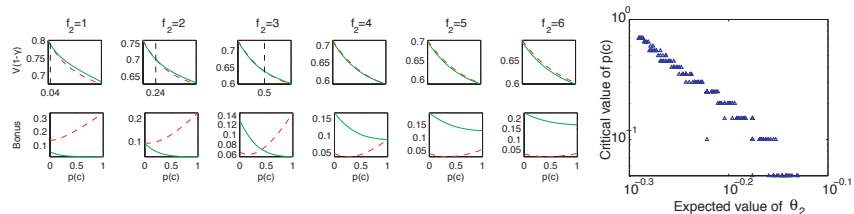

Figure 3: Value of arms as a function of coupling. The priors are uniform ($\alpha_j = \beta_j = 1$, $1 \leq j \leq 3$), the evidence for arm 1 remains fixed for all cases ($s_1 = 1, f_1 = 0$), and successes of arm 2 remains fixed as well ($s_2 = 5$). Failures for arm 2 ($f_2$) vary from 1 to 6 . **Upper left**: Belief that arms are coupled ($p(c)$) versus reward per unit time ($V(1 - \gamma)$, where $V$ is the value) of arm 1 (dashed line) and arm 2 (solid line). In all cases, an independent model would choose arm 1 to pull. Vertical line shows the critical coupling belief value where the structure learning model switches to exploitative behavior. **Lower left**: Exploratory bonus ($V(1 - \gamma) - r$, where $r$ is the expected reward) for each arm. **Right panel**: Critical coupling belief values for exploitative behavior vs. the expected probability of reward of arm 2. Individual points correspond to different information states (successes and failures on both arms).

## 5   Human Experiments

Each of 16 subjects ran on 32 bandit tasks, a block of 16 in a independent environment and a block of 16 coupled. Within blocks, the presentation order was randomized, and the order of the coupled environment was randomized accross subjects. On average each task required 48 pulls. For independent environment, the subjects made 1194 choices across the 16 tasks, and 925 for the coupled environment.

Each arm is shown in the screen as a slot machine. Subjects pull a machine by pressing a key in the keyboard. When pulled, an animation of the lever is shown, 200 msec later the reward appears in the machine's screen, and a sound mimicking dropping coins lasts proportionally to the amount gathered. We provide several cues, some redundant, to help subjects keep track of previous rewards. At the top, the machine shows the number of pulls, total reward, and average reward per pull so far. Instead of binary rewards 0 and 1, the task presented 0 and 100. The machine's screen changes the color according to the average reward, from red (zero points), through yellow (fifty points), and green (one hundred points). The machine's total reward is shown as a pile of coins underneath it. The total score, total pulls, and rankings within a game were presented.

## 6   Results

We analyze how task uncertainty affects decisions by comparing human behavior to that of the optimal model and models that assume a structure. For each agent, be human or not, we compute the (empirical) probability that it selects the oracle-best action versus the optimal belief that a block of tasks is coupled. The idea behind this measure is to show how the belief on task structure changes the behavior and which of the models better captures human behavior.

We run 1000 agents for each of the models with task uncertainty (*optimal model*), assumed coupled reward task (*coupled model*), and assumed independent reward task (*independent model*) under the same conditions that subjects faced on both the blocks of coupled and independent tasks. And for each of the decisons of these models and the 33904 decisions performed by the 16 subjects, we compute the optimal belief on coupling according to our model and bin the proportion of times the agent chooses the (oracle) best arm according to this belief. The results are summarized in Figure 4. The independent model tends to perform equally well on both coupled and independent reward tasks. The coupled model tends to perform well only in the coupled task and worse in the independent tasks. As expected, the optimal model has better overall performance, but does not perform better than models with fixed task structure—in their respective tasks—because it pays the price of learning early in the block. The optimal model behaves like a mixture between the coupled and independent model. Human behavior is much better captured by the optimal model (Figure

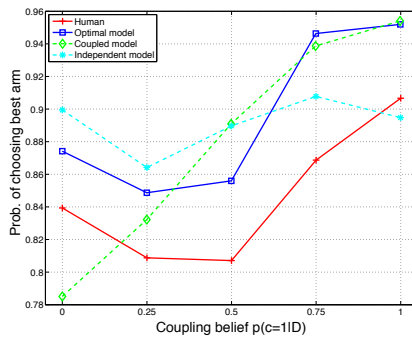

Figure 4: Effect of coupling on behavior. For each of the decisions of subjects and simulated models under the same conditions, we compute the optimal belief on coupling according to the model proposed in this paper and bin the proportion of times an agent chooses the (oracle) best arm according to this belief. This plot represents the empirical probability that an agent would pick the best arm at a given belief on coupling.

4). This is evidence that human behavior shares the characteristics of the optimal model, namely, it contains task uncertainty and exploit the knowledge of the task structure to maximize its gains. The gap in performance that exists between the optimal model and humans may be explained by memory limitations or more complicated task structures being entertained by subjects. Because the subjects are not told the coupling state of the environment and the arms appear as separate options we conclude that people are capable of learning and exploiting task structure. Together these results suggest that structure learning may play a significant role in explaining differences between human behavior and previous normative predictions.

## 7    Conclusions and future directions

We have provided evidence that structure learning may be an important missing piece in evaluating human sequential decision making. The idea of modeling sequential decision making under uncertainty as a structure learning problem is a natural extension of previous work on structure learning in Bayesian models of cognition [17, 18] and animal learning [19] to sequential decision making problems under uncertainty. It also extends previous work on Bayesian approaches to modeling sequential decision making in the multi-armed bandit [20] by adding structure learning. It is important to note that we have intentionally focused on reward structure, ignoring issues involving dependencies across trials. Clearly reward structure learning must be integrated with learning about temporal dependencies [21].

Although we focused on learning coupling between arms, there are other kinds of reward structure learning that may account for a broad variety of human decision making performance. In particular, allowing dependence between the probability of reward at a site and previous actions can produce large changes in decision making behavior. For instance, in a "foraging" model where reward is collected from a site and probabilistically replenished, optimal strategies will produce choice sequences that alternate between reward sites. Thus uncertainty about the independence of reward on previous actions can produce a continuum of behavior, from maximization to probability matching. Note that structure learning explanations for probability matching is significantly different than explanations based on reinforcing previously successful actions (the "law of effect") [22]. Instead of explaining behavior in terms of the idiosyncracies of a learning rule, structure learning constitutes a fully rational response to uncertainty about the causal structure of rewards in the environment. We intend to test the predictive power of a range of structure learning ideas on experimental data we are currently collecting. Our hope is that, by expanding the range of normative hypotheses for human decision-making, we can begin to develop more principled accounts of human sequential decision-making behavior.

**Acknowledgements**

The work was supported by NIH NPCS 1R90 DK71500-04, NIPS 2008 Travel Award, CONICYT-FIC-World Bank 05-DOCFIC-BANCO-01, ONR MURI N 00014-07-1-0937, and NIH EY02857.

# References

[1] Pascal Poupart, Nikos Vlassis, Jesse Hoey, and Kevin Regan. An analytic solution to discrete bayesian reinforcement learning. In *23rd International Conference on Machine Learning*, Pittsburgh, Penn, 2006.

[2] Richard Ernest Bellman. *Dynamic programming*. Princeton University Press, Princeton, 1957.

[3] Noah Gans, George Knox, and Rachel Croson. Simple models of discrete choice and their performance in bandit experiments. *Manufacturing and Service Operations Management*, 9(4):383–408, 2007.

[4] C.M. Anderson. *Behavioral Models of Strategies in Multi-Armed Bandit Problems*. PhD thesis, Pasadena, CA., 2001.

[5] Jeffrey Banks, David Porter, and Mark Olson. An experimental analysis of the bandit problem. *Economic Theory*, 10(1):55–77, 1997.

[6] R. J. Meyer and Y. Shi. Sequential choice under ambiguity: Intuitive solutions to the armed-bandit problem. *Management Science*, 41:817–83, 1995.

[7] N Vulkan. An economist's perspective on probability matching. *Journal of Economic Surveys*, 14:101–118, 2000.

[8] Yvonne Brackbill and Anthony Bravos. Supplementary report: The utility of correctly predicting infrequent events. *Journal of Experimental Psychology*, 64(6):648–649, 1962.

[9] W Edwards. Probability learning in 1000 trials. *Journal of Experimental Psychology*, 62:385–394, 1961.

[10] W Edwards. Reward probability, amount, and information as determiners of sequential two-alternative decisions. *J Exp Psychol*, 52(3):177–88, 1956.

[11] E. Fantino and A Esfandiari. Probability matching: Encouraging optimal responding in humans. *Canadian Journal of Experimental Psychology*, 56:58 – 63, 2002.

[12] Timothy E J Behrens, Mark W Woolrich, Mark E Walton, and Matthew F S Rushworth. Learning the value of information in an uncertain world. *Nat Neurosci*, 10(9):1214–1221, 2007.

[13] N. D. Daw, J. P. O'Doherty, P. Dayan, B. Seymour, and R. J. Dolan. Cortical substrates for exploratory decisions in humans. *Nature*, 441(7095):876–879, 2006.

[14] JS Banks and RK Sundaram. A class of bandit problems yielding myopic optimal strategies. *Journal of Applied Probability*, 29(3):625–632, 1992.

[15] John Gittins and You-Gan Wang. The learning component of dynamic allocation indices. *The Annals of Statistics*, 20(2):1626–1636, 1992.

[16] J. C. Gittins and D. M. Jones. A dynamic allocation index for the sequential design of experiments. *Progress in Statistics*, pages 241–266, 1974.

[17] Joshua B. Tenenbaum, Thomas L. Griffiths, and Charles Kemp. Theory-based bayesian models of inductive learning and reasoning. *Trends in Cognitive Sciences*, 10(7):309–318, 2006.

[18] Joshua B. Tenenbaum and Thomas L. Griffiths. Structure learning in human causal induction. *NIPS 13*, pages 59–65, 2000.

[19] A. C. Courville, N. D. Daw, G. J. Gordon, and D. S. Touretzky. Model uncertainty in classical conditioning. *Advances in Neural Information Processing Systems*, (16):977–986, 2004.

[20] Daniel Acuna and Paul Schrater. Bayesian modeling of human sequential decision-making on the multi-armed bandit problem. In *CogSci*, 2008.

[21] Michael D. Lee. A hierarchical bayesian model of human decision-making on an optimal stopping problem. *Cognitive Science: A Multidisciplinary Journal*, 30:1 – 26, 2006.

[22] Ido Erev and Alvin E. Roth. Predicting how people play games: Reinforcement learning in experimental games with unique, mixed strategy equilibria. *The American Economic Review*, 88(4):848–881, 1998.

